# Partially Observable SDE Models for Image Sequence Recognition Tasks

**Javier R. Movellan**
Institute for Neural Computation
University of California San Diego

**Paul Mineiro**
Department of Cognitive Science
University of California San Diego

**R. J. Williams**
Department of Mathematics
University of California San Diego

## Abstract

This paper explores a framework for recognition of image sequences using partially observable stochastic differential equation (SDE) models. Monte-Carlo importance sampling techniques are used for efficient estimation of sequence likelihoods and sequence likelihood gradients. Once the network dynamics are learned, we apply the SDE models to sequence recognition tasks in a manner similar to the way Hidden Markov models (HMMs) are commonly applied. The potential advantage of SDEs over HMMS is the use of continuous state dynamics. We present encouraging results for a video sequence recognition task in which SDE models provided excellent performance when compared to hidden Markov models.

## 1  Introduction

This paper explores a framework for recognition of image sequences using partially observable stochastic differential equations (SDEs). In particular we use SDE models of low-power non-linear RC circuits with a significant thermal noise component. We call them diffusion networks. A diffusion network consists of a set of $n$ nodes coupled via a vector of adaptive impedance parameters $\lambda$ which are tuned to optimize the network's behavior. The temporal evolution of the $n$ nodes defines a continuous stochastic process $X$ that satisfies the following Itô SDE:

$$dX(t) = \mu(X(t), \lambda)dt + \sigma \, dB(t), \qquad (1)$$
$$X(0) \sim \nu, \qquad (2)$$

where $\nu$ represents the (stochastic) initial conditions and $B$ is standard Brownian motion. The drift is defined by a non-linear RC charging equation

$$\mu_j(X(t), \lambda) = \frac{1}{\kappa_j} \left( \xi_j + \bar{X}_j(t) - \frac{1}{\rho_j} X_j(t) \right), \quad \text{for } j = 1, \cdots, n, \qquad (3)$$

where $\mu_j$ is the drift of unit $j$, i.e., the $j^{th}$ component of $\mu$. Here $X_j$ is the internal potential at node $j$, $\kappa_j > 0$ is the input capacitance, $\rho_j$ the node resistance, $\xi_j$ a

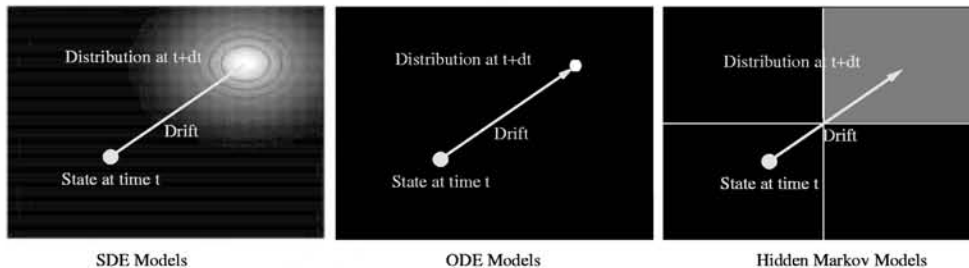

Figure 1: An illustration of the differences between stochastic differential equation models (SDE), ordinary differential equation models (ODE) and Hidden Markov Models (HMM). In ODEs the the state dynamics are continuous and deterministic. In SDEs the state dynamics are continuous and stochastic. In HMMs the state dynamics are discrete and probabilistic.

constant input current to the unit, $\bar{X}_j$ the net electrical current input to the node,

$$\bar{X}_j(t) = \sum_{m=1}^{n} w_{j,m}\,\varphi(X_m(t)), \quad \text{for } j = 1, \cdots, n, \tag{4}$$

$$\varphi(x) = \frac{1}{1 + e^{-x}}, \text{ for all } x \in \mathbb{R}, \tag{5}$$

where $\varphi$ the input-output characteristic amplification, and $1/w_{j,m}$ is the impedance between the output $X_m$ and the node $j$. Intuition for equation (3) can be achieved by thinking of it as the limit of a discrete time stochastic difference equation,

$$X(t + \Delta t) = X(t) + \mu(X(t), \lambda)\Delta t + \sigma\sqrt{\Delta t}Z(t), \tag{6}$$

where the $Z(t)$ is an n-dimensional vector of independent standard Gaussian random variables. For a fixed state at time $t$ there are two forces controlling the change in activation: the drift, which is deterministic, and the dispersion which is stochastic (see Figure 1). This results in a distribution of states at time $t + \Delta t$. As $\Delta t$ goes to zero, the solution to the difference equation (6) converges to the diffusion process defined in (3).

Figures 1 and 2 shows the relationship between SDE models and other approaches in the neural network and the stochastic filtering literature. The main difference between ODE models, like standard recurrent neural networks, and SDE models is that the first has deterministic dynamics while the second has probabilistic dynamics. The two approaches are similar in that the states are continuous. The main difference between HMMs and SDEs is that the first have discrete state dynamics while the second have continuous state dynamics. The main similarity is that both are probabilistic. Kalman filters are linear SDE models. If the impedance matrix is symmetric and the network is given enough time to approximate stochastic equilibrium, diffusion network behave like continuous Boltzmann machines (Ackley, Hinton & Sejnowski, 1985). If the network is discretized in state and time it becomes a standard HMM. Finally, if the dispersion constant is set to zero the network behaves like a deterministic recurrent neural network.

In order to use of SDE models we need a method for finding the likelihood and the likelihood gradient of observed sequences.

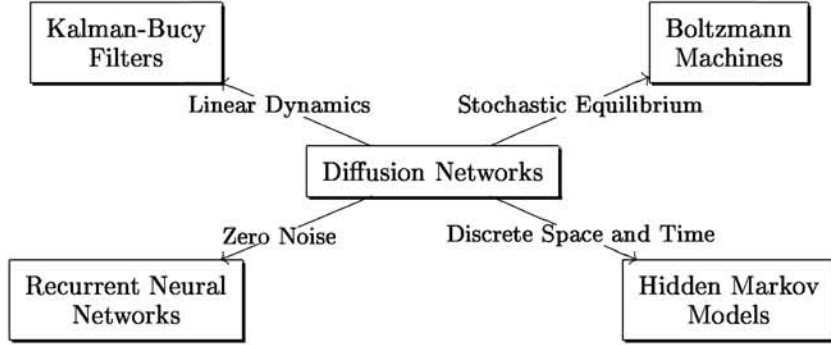

Figure 2: Relationship between diffusion filters and other approaches in the neural network and stochastic filtering literature.

## 2 Observed sequence likelihoods

We regard the first $d$ components of an SDE model as *observable* and denote them by $O$. The last $n - d$ components are denoted by $H$ and named *unobservable* or *hidden*. Hidden components are included for modeling non-Markovian dependencies in the observable components. Let $\Omega_o$, $\Omega_h$ be the outcome spaces for the observable and hidden processes. Let $\Omega = \Omega_o \times \Omega_h$ the joint outcome space. Here each outcome $\omega$ is a continuous path $\omega : [0, T] \to \mathbb{R}^n$. For each $\omega \in \Omega$, we write $\omega = (\omega_o, \omega_h)$, where $\omega_o$ represents the observable dimensions of the path and $\omega_h$ the hidden dimensions. Let $Q^\lambda(A)$ represent the probability that a network with parameter $\lambda$ generates paths in the set $A$, $Q_o^\lambda(A_o)$ the probability that the observable components generate paths in $A_o$ and $Q_h^\lambda(A_h)$ the probability that the hidden components generate paths in $A_h$. To apply the familiar techniques of maximum likelihood and Bayesian estimation we use as reference the probability distribution of a diffusion network with zero drift, i.e., the paths generated by this network are Brownian motion scaled by $\sigma$. We denote such reference distribution as $R$, its observable and hidden components as $R_o$, $R_h$. Using Girsanov's theorem (Karatzas & Shreve, 1991, p. 303) we have that

$$L_o^\lambda(\omega_o) = \frac{dQ_o^\lambda}{dR_o}(\omega_o) = \int_{\Omega_h} L_{o,h}^\lambda(\omega_o, \omega_h) \, dR_h(\omega_h), \ \omega_o \in \Omega_o, \tag{7}$$

where

$$L_{o,h}^\lambda(\omega) = \frac{dQ^\lambda}{dR}(\omega) = \exp\left\{ \frac{1}{\sigma^2} \int_0^T \mu(\omega(t), \lambda) \cdot d\omega(t) - \frac{1}{2\sigma^2} \int_0^T |\mu(\omega(t), \lambda)|^2 dt \right\}. \tag{8}$$

The first integral in (8) is an Itô stochastic integral, the second is a standard Lebesgue integral. The term $L_o^\lambda$ is a Radon-Nikodym derivative that represents the probability density of $Q_o^\lambda$ with respect to $R_o$. For a fixed path $\omega_o$ the term $L_o^\lambda(\omega_o)$ is a likelihood function of $\lambda$ that can be used for Maximum likelihood estimation. To obtain the likelihood gradient, we differentiate (7) which yields

$$\nabla_\lambda \log L_o^\lambda(\omega_o) = \int_{\Omega_h} L_{h|o}^\lambda(\omega_h \mid \omega_o) \nabla_\lambda \log L_{o,h}^\lambda(\omega_o, \omega_h) \, dR_h(\omega_h), \tag{9}$$

where

$$L_{h|o}^{\lambda}(\omega_h \mid \omega_o) = \frac{L_{o,h}^{\lambda}(\omega_o, \omega_h)}{L_o^{\lambda}(\omega_o)}, \tag{10}$$

$$\nabla_{\lambda} \log L_{o,h}^{\lambda}(\omega) = \frac{1}{\sigma^2} \int_0^T J(t, \omega(t), \lambda) \cdot dI^{\lambda}(t, \omega), \tag{11}$$

$$J_{j,k}(t, \omega, \lambda) = \frac{\partial \mu_k(\omega(t), \lambda)}{\partial \lambda_j}, \tag{12}$$

$$\tag{13}$$

and $I^{\lambda}$ is the joint innovation process

$$I^{\lambda}(t, \omega) = \omega(t) - \omega(0) - \int_0^t \mu(\omega(u), \lambda) \ du. \tag{14}$$

## 2.1 Importance sampling

The likelihood of observed paths (7), and the gradient of the likelihood (9) require averaging with respect to the distribution of hidden paths $R_h$. We estimate these averages using an importance sampling in the space of sample paths. Instead of sampling from $R_h$ we sample from a distribution that weights more heavily regions where $L_{o,h}^{\lambda}$ is large. Each sample is then weighted by the density of the sampling distribution with respect to $R_h$. This weighting function is commonly known as the importance function in the Monte-Carlo literature (Fishman, 1996, p. 257). In particular for each observable path $\omega_o$ we let the sampling distribution $S_h^{\lambda, \omega_o}$ be the probability distribution generated by a diffusion network with parameter $\lambda$ which has been forced to exhibit the path $\omega_o$ over the observable units. The approach reminiscent of the technique of teacher forcing from deterministic neural networks. In practice, we generate i.i.d. sampled hidden paths $\{h^{(i)}\}_{i=1}^m$ from $S_h^{\lambda, \omega_o}$ by numerically simulating a diffusion network with the observable units forced to exhibit the path $\omega_o$ these hidden paths are then weighted by the density of $S_h^{\lambda, \omega_o}$ with respect to $R_h$, which acts as a Monte-Carlo importance function

$$\frac{dR_h}{dS_h^{\lambda, \omega_o}}(\omega_h) = \exp - \left\{ \frac{1}{\sigma^2} \int_0^T \mu_h((\omega_o(t), \omega_h(t)), \lambda) \cdot d\omega_h(t) \right.$$
$$\left. - \frac{1}{2\sigma^2} \int_0^T |\mu_h((\omega_o(t), \omega_h(t)), \lambda)|^2 dt \right\}. \tag{15}$$

In practice we have obtained good results with $m$ in the order of 20, i.e., we sample 20 hidden sequences per observed sequence. One interesting property of this approach is that the sampling distributions $S_h^{\lambda, \omega_o}$ change as learning progresses, since they depend on $\lambda$.

Figure 3 shows results of a computer simulation in which a 2 unit network was trained to oscillate. We tried an oscillation pattern because of its relevance for the application we explore in a later section, which involves recognizing sequences of lip movements. The figure shows the "training" path and a couple of sample paths, one obtained with the $\sigma$ parameter set to 0, and one with the parameter set to 0.5.

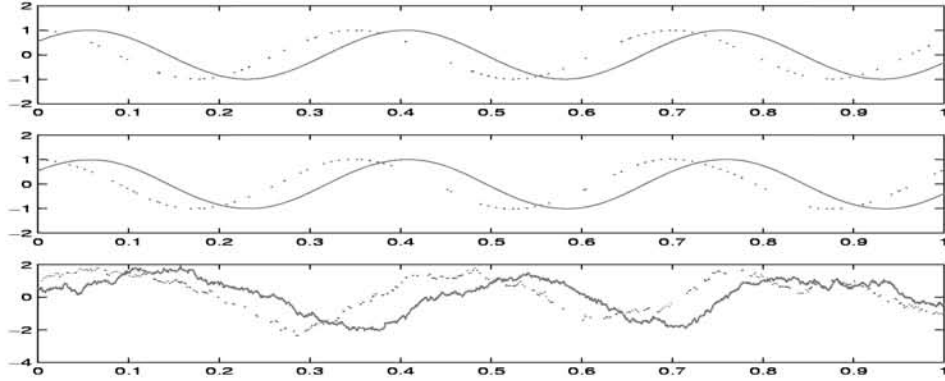

Figure 3: Training a 2 unit network to maximize the likelihood of a sinusoidal path. The top graph shows the training path. It consists of two sinusoids out of phase each representing the activation of the two units in the network. The center graph shows a sample path obtained after training the network and setting $\sigma = 0$, i.e., no noise. The bottom graph shows a sample path obtained with $\sigma = 0.5$.

## 3   Recognizing video sequences

In this section we illustrate the use of SDE models on a sequence classification task of reasonable difficulty with a body of realistic data. We chose this task since we know of SDE models used for tracking problems but know of no SDE models used for sequence recognition tasks. The potential advantage of SDEs over more established approaches such as HMMs is that they enforce continuity constraints, an aspect that may be beneficial when the actual signals are better described using continuous state dynamics. We compared a diffusion network approach with classic hidden Markov model approaches.

We used Tulips1 (Movellan, 1995), a database consisting of 96 movies of 9 male and 3 female undergraduate students from the Cognitive Science Department at the University of California, San Diego. For each student two sample utterances were taken for each of the digits "one" through "four". The database is available at http://cogsci.ucsd.edu. We compared the performance of diffusion networks and HMMs using two different image processing techniques (contours and contours plus intensity) in combination with 2 different recognition engines (HMMs and diffusion networks). The image processing was performed by Luettin and colleagues (Luettin, 1997). They employ point density models, where each lip contour is represented by a set of points; in this case both the inner and outer lip contour are represented, corresponding to Luettin's double contour model. The dimensionality of the representation of the contours is reduced using principal component analysis. For the work presented here 10 principal components were used to approximate the contour, along with a scale parameter which measured the pixel distance between the mouth corners; associated with each of these 11 parameters was a corresponding "delta component", the left-hand temporal difference of the component (defined to be zero for the first frame). In this manner a total of 22 parameters were used to represent lip contour information for each still frame. These 22 parameters were represented using diffusion networks with 22 observation units, one per parameter value. We also tested the performance of a representation that used intensity information in addition to contour shape information. This approach used 62 parameters, which were represented using diffusion networks with 62 observation units.

| Approach | Correct Generalization |
|---|---|
| Best HMM, shape information only | 82.3% |
| Best diffusion network, shape information only | 85.4% |
| Untrained human subjects | 89.9% |
| Best HMM, shape and intensity information | 90.6% |
| Best diffusion network, shape and intensity information | 91.7% |
| Trained human subjects | 95.5% |

Table 1: Average generalization performance on the Tulips1 database. Shown in order are the performance of the best performing HMM from (Luettin et al., 1996), which uses only shape information, the best diffusion network obtained using only shape information, the performance of untrained human subjects (Movellan, 1995), the HMM from Luettin's thesis (Luettin 1997) which uses both shape and intensity information, the best diffusion network obtained using both shape and intensity information, and the performance of trained human lipreaders (Movellan, 1995).

We independently trained 4 diffusion networks, to approximate the distributions of lip-contour trajectories of each of the four words to be recognized, i.e., the first network was trained with examples of the word "one", and the last network with examples of the word "four". Each network had the same number of nodes, and the drift of each network was given by (3) with $\kappa_i = 1$, $\frac{1}{\rho_i} = 0$ for all units, and $\xi$ being part of the adaptive vector $\lambda$. Thus, $\lambda = (\xi_1, \cdots, \xi_n, w_{1,1}, w_{1,2}, \cdots w_{n,n})'$. The number of hidden units was varied from one to 5. We obtained optimal results with 4 hidden units. The initial state of the hidden units was set to $(1, \ldots, 1)$ with probability 1, and $\sigma$ was set to 1 for all networks. The diffusion network dynamics were simulated using a forward-Euler technique, i.e., equation (1) is approximated in discrete time using (6). In our simulations we set $\Delta t = 1/30$ seconds, the time between video frame samples. Each diffusion network was trained with examples of one of the 4 digits using the cost function

$$\hat{\Phi}(\lambda) = \sum_i \log \hat{L}_o^\lambda(y^{(i)}) - \frac{1}{2}\alpha|\lambda|^2, \tag{16}$$

where $\{y^{(i)}\}$ are samples from the desired empirical distribution $P_0$ and $\alpha$ is the strength of a Gaussian prior on the network parameters. Best results were obtained with diffusion networks with 4 hidden units. The log-likelihood gradients were estimated using the importance sampling approach with $m = 20$, i.e., we generated 20 hidden sample paths per observed path. With this number of samples training took about 10 times longer with diffusion networks than with HMMs. At test time, computation of the likelihood estimates was very fast and could have been done in real time using a fast Pentium II.

The generalization performance was estimated using a jacknife (one-out) technique: we trained on all subjects but one, which is used for testing. The process is repeated leaving a different subject out every time. Results are shown in Table 1. The table includes HMM results reported by Luettin (1997), who tried a variety of HMM architectures and reported the best results obtained with them. The only difference between Luettin's approach and our approach is the recognition engine, which was a bank of HMMs in his case and a bank of diffusion networks in our case. If anything we were at a disadvantage since the image representations mentioned above were optimized by Luettin to work best with HMMs.

In all cases the best diffusion networks outperformed the best HMMs reported in the literature using exactly the same visual preprocessing. In all cases diffusion net-

works outperformed HMMs. The difference in performance was not large. However obtaining even a 1% increment in performance on this database is very difficult.

## 4 Discussion

While we presented results for a video sequence recognition task, the same framework can be used for tasks such as sequence recognition, object tracking and sequence generation. Our work was inspired by the rich literature on continuous stochastic filtering and stochastic neural networks. The idea was to combine the versatility of recurrent neural networks and the well known advantages of stochastic modeling approaches. The continuous-time nature of the networks is convenient for data with dropouts or variable sample rates, since the models we use define all the finite dimensional distributions. The continuous-state representation is well suited to problems involving inference about continuous unobservable quantities, as in visual tracking tasks. Since these networks enforce continuity constraints in the observable paths they may not have the well known problems encountered when HMMs are used as generative models of continuous sequences.

We have presented encouraging results on a realistic sequence recognition task. However more work needs to be done, since the database we used is relatively small. At this point the main disadvantage of diffusion networks relative to conventional hidden Markov models is training speed. The diffusion networks used here were approximately 10 times slower to train than HMMs. Fortunately the Monte Carlo approximations employed herein, which represent the bulk of the computational burden, lend themselves to parallel and hardware implementations. Moreover, once a network is trained, the computation of the density functions needed in recognition tasks can be done in real time.

We are exploring applications of diffusion networks to stochastic filtering problems (e.g., contour tracking) and sequence generation problems, not just sequence recognition problems. Our work shows that diffusion networks may be a feasible alternative to HMMs for problems in which state continuity is advantageous. The results obtained for the visual speech recognition task are encouraging, and reinforce the possibility that diffusion networks may become a versatile tool for a very wide variety of continuous signal processing tasks.

## References

Ackley, D. H., Hinton, G. E., & Sejnowski, T. (1985). A Learning Algorithm for Boltzmann Machines. *Cognitive Science*, *9*(2), 147–169.

Fishman, G. S. (1996). *Monte Carlo Sampling: Concepts Algorithms and Applications*. New York: Sprienger-Verlag.

Karatzas, I. & Shreve, S. (1991). *Brownian Motion and Stochastic Calculus*. Springer.

Luettin, J. (1997). *Visual Speech and Speaker Recognition*. PhD thesis, University of Sheffield.

Movellan, J. (1995). Visual Speech Recognition with Stochastic Neural Networks. In G. Tesauro, D. Touretzky, & T. Leen (Eds.), *Advances in Neural Information Processing Systems*, volume 7. MIT Press.

Oksendal, B. (1992). *Stochastic Differential Equations*. Berlin: Springer Verlag.
